# On Transductive Regression

**Corinna Cortes**
Google Research
76 Ninth Avenue
New York, NY 10011
corinna@google.com

**Mehryar Mohri**
Courant Institute of Mathematical Sciences
and Google Research
251 Mercer Street
New York, NY 10012
mohri@cs.nyu.edu

## Abstract

In many modern large-scale learning applications, the amount of unlabeled data far exceeds that of labeled data. A common instance of this problem is the *transductive* setting where the unlabeled test points are known to the learning algorithm. This paper presents a study of regression problems in that setting. It presents *explicit* VC-dimension error bounds for transductive regression that hold for all bounded loss functions and coincide with the tight classification bounds of Vapnik when applied to classification. It also presents a new transductive regression algorithm inspired by our bound that admits a primal and kernelized closed-form solution and deals efficiently with large amounts of unlabeled data. The algorithm exploits the position of unlabeled points to locally estimate their labels and then uses a global optimization to ensure robust predictions. Our study also includes the results of experiments with several publicly available regression data sets with up to 20,000 unlabeled examples. The comparison with other transductive regression algorithms shows that it performs well and that it can scale to large data sets.

## 1 Introduction

In many modern large-scale learning applications, the amount of unlabeled data far exceeds that of labeled data. Large amounts of digitized data are widely available but the cost of labeling is often prohibitive since it typically requires human assistance. Semi-supervised learning or transductive inference leverage unlabeled data to achieve better predictions and are thus particularly relevant to modern applications. Semi-supervised learning consists of using both labeled and unlabeled data to find a hypothesis that accurately labels unseen examples. Transductive inference uses the same information but only aims at predicting the labels of the known unlabeled examples.

This paper deals with regression problems in the transductive setting, which arise in a variety of contexts. This may be to predict the real-valued labels of the nodes of a known graph in computational biology, or the scores associated to known documents in information extraction problems. The problem of transduction inference was originally formulated and analyzed by Vapnik [1982] who described it as a simpler task than the traditional induction treated in machine learning. A number of recent publications have dealt with the topic of transductive inference [Vapnik, 1998, Joachims, 1999, Bennett and Demiriz, 1998, Chapelle et al., 1999, Graepel et al., 1999, Schuurmans and Southey, 2002, Corduneanu and Jaakkola, 2003, Zhu et al., 2004, Lanckriet et al., 2004, Derbeko et al., 2004, Belkin et al., 2004, Zhou et al., 2005]. But, with the exception of [Chapelle et al., 1999], [Schuurmans and Southey, 2002], and [Belkin et al., 2004], this work has primarily dealt with classification problems.

We present a specific study of transductive regression. We give new error bounds for transductive regression that hold for all bounded loss functions and coincide with the tight classification bounds of Vapnik [1998] when applied to classification. Our results also include explicit VC-dimension bounds for transductive regression. This contrasts with the original regression bound given by Vapnik [1998] which assumes a specific condition of global regularity on the class of functions and is based on a complicated and implicit function of the samples sizes and the confidence parameter. As stated by Vapnik [1998], this function must be "tabulated by a computer".

We also present a new algorithm for transductive regression inspired by our bound which first exploits the position of unlabeled points to locally estimate their labels, and then uses a global optimization to ensure robust predictions. We show that our algorithm admits both a primal and a kernelized closed-form solution. Existing algorithms for the transductive setting require the inversion of a matrix whose dimension is either the total number of unlabeled and labeled examples [Belkin et al., 2004], or the total number of unlabeled examples [Chapelle et al., 1999]. This may be prohibitive for many real-world applications with very large amounts of unlabeled examples. One of the original motivations for our work was to design algorithms dealing precisely with such situations. When the dimension of the feature space $N$ is not too large, our algorithm provides a very efficient solution whose cost is dominated by the construction and inversion of an $N \times N$-matrix. Similarly, when the number of training points $m$ is small compared to the number of unlabeled points, using an empirical kernel map, our algorithm requires only constructing and inverting an $m \times m$-matrix.

Our study also includes the results of our experiments with several publicly available regression data sets with up to 20,000 unlabeled examples, limited only by the size of the data sets. We compared our algorithm with those of Belkin et al. [2004] and Chapelle et al. [1999], which are among the very few algorithms described in the literature dealing specifically with the problem of transductive regression. The results show that our algorithm performs well in several data sets compared to these algorithms and that it can scale to large data sets.

The paper is organized as follows. Section 2 describes in more detail the transductive regression setting we are studying. New generalization error bounds for transductive regression are presented in Section 3. Section 4 describes and analyzes both the primal and dual versions of our algorithm and the experimental results of our study are reported in Section 5.

## 2  Definition of the Problem

Assume that a full sample $\mathcal{X}$ of $m + u$ examples is given. The learning algorithm further receives the labels of a random subset of $\mathcal{X}$ of size $m$ which serves as a training sample:

$$(x_1, y_1), \ldots, (x_m, y_m) \in \mathcal{X} \times \mathbb{R}. \tag{1}$$

The remaining $u$ unlabeled examples, $x_{m+1}, \ldots, x_{m+u} \in \mathcal{X}$, serve as test data. The learning problem that we consider consists of predicting accurately the labels $y_{m+1}, \ldots, y_{m+u}$ of the test examples. No other test examples will ever be considered. This is a *transduction regression* problem [Vapnik, 1998].[1] It differs from the standard (*induction*) regression estimation problem by the fact that the learning algorithm is given the unlabeled test examples beforehand. Thus, it may exploit that information and achieve a better result than via the standard induction.

In what follows, we consider a hypothesis space $H$ of real-valued functions for regression estimation. For a hypothesis $h \in H$, we denote by $R_0(h)$ its mean squared error on the full sample, by $\widehat{R}(h)$ its error on the training data, and by $R(h)$ the error of $h$ on the test examples:

$$R_0(h) = \frac{1}{m+u} \sum_{i=1}^{m+u} (h(x_i) - y_i)^2 \quad \widehat{R}(h) = \frac{1}{m} \sum_{i=1}^{m} (h(x_i) - y_i)^2 \quad R(h) = \frac{1}{u} \sum_{i=m+1}^{m+u} (h(x_i) - y_i)^2.$$
$$\tag{2}$$

For convenience, we will sometimes denote by $y_x = y_i$ the label of a point $x = x_i \in \mathcal{X}$.

## 3  Transductive Regression Generalization Error

This section presents explicit generalization error bounds for transductive regression.

Vapnik [1998] introduced and analyzed the problem of transduction and presented transductive inference bounds for both classification and regression. His regression bound assumes however a specific regularity condition on the hypothesis functions leading in particular to a surprising bound where no error on the training data implies zero generalization error. The bound has the multiplicative form: $R(h) \le \Omega(m, u, d, \delta)\widehat{R}(h)$, where $d$ is the VC-dimension of the class of hypotheses used and $\delta$ is the confidence parameter. Furthermore, for certain values of the parameters, for example larger $d$s or smaller $\delta$s, $\Omega$ becomes infinite and the bound is ineffective [Vapnik, 1998, page 349]. $\Omega$ is also based on a complicated and implicit function of $m$, $u$, and $\delta$, which makes its interpretation difficult. For example, it is hard to analyze the asymptotic behavior of the bound for large $u$.

Instead, our bounds simply hold for general bounded loss functions and, when applied to classification, coincide with the tight classification bounds of Vapnik [1998]. Our results also include explicit VC-dimension bounds for transductive regression. To the best of our knowledge, these are the first general explicit bounds for transductive regression.

Our first bound uses the function $\bar{\Gamma}(\epsilon, k)$ defined as follows. Let $\Gamma(\epsilon, k)$ be defined by:

$$\forall \epsilon \geq 0, \forall k \in \mathbb{N}, u\epsilon \leq k \leq m(1 - \epsilon) + u, \quad \Gamma(\epsilon, k) = \sum_{r \in I(m, u, \epsilon)} \frac{\binom{k}{r}\binom{m+u-k}{m-r}}{\binom{m+u}{m}}, \quad (3)$$

where $I(m, u, k, \epsilon)$ is the set of integers $r$ such that: $\frac{k-r}{u} - \frac{r}{m} > \epsilon$ and $\max(0, k - u) \leq r \leq \min(m, k)$. $\Gamma(\epsilon, k)$ represents the probability of observing a difference in error rate of more than $\epsilon$ between the training and test set when the total number of errors is $k$ (see [Cortes and Mohri, 2006]). Then $\bar{\Gamma}$ is defined as $\bar{\Gamma}(\epsilon) = \max_k \Gamma(\sqrt{\frac{k}{m+u}}\epsilon, k)$. $\bar{\Gamma}$ is used in the transductive classification bound of Vapnik [1998] (see [Cortes and Mohri, 2006][Theorem 2]). [Cortes and Mohri, 2006][Corollary 2] gives an upper bound on $\bar{\Gamma}$.

For any subset $\mathcal{X}' \subseteq \mathcal{X}$, any non-negative real number $t \geq 0$, and hypothesis $h \in H$, let $\Theta(h, t, \mathcal{X}')$ denote the fraction of the points $x_i \in \mathcal{X}'$, $i = 1, \ldots, k$, such that $(h(x_i) - y_i)^2 - t > 0$. Thus, $\Theta(h, t, \mathcal{X}')$ represents the error rate over the sample $\mathcal{X}'$ of the classifier that associates to a point $x$ the value zero if $(h(x) - y_x)^2 \leq t$, one otherwise.

Two classifiers associated in this way to $\Theta(h, t, \mathcal{X})$ and $\Theta(h', t', \mathcal{X})$ can be viewed as equivalent if they label $\mathcal{X}$ in an identical way. Since $\mathcal{X}$ is finite, there is a finite number of equivalence classes of such classifiers, we will denote that number by $\mathcal{N}(m + u)$.

**Theorem 1** *Let $\delta > 0$, and let $\epsilon_0 > 0$ be the minimum value of $\epsilon$ such that $\mathcal{N}(m + u)\bar{\Gamma}(\epsilon) \leq \delta$, and assume that the loss function is bounded: for all $h \in H$ and $x \in \mathcal{X}$, $(h(x) - y_x)^2 \leq B^2$, where $B \in \mathbb{R}_+$. Then, with probability at least $1 - \delta$, for all $h \in H$,*

$$R(h) \leq \widehat{R}(h) + \frac{u\epsilon_0^2 B^2}{2(m+u)} + \epsilon_0 B \sqrt{\widehat{R}(h) + \left(\frac{u\epsilon_0 B}{2(m+u)}\right)^2}. \quad (4)$$

*Proof.* For any $h \in H$, let $R_1(h)$ be defined by:

$$R_1(h) = \int_0^{B^2} \sqrt{\Theta(h, t, \mathcal{X})}\, dt. \quad (5)$$

By the Cauchy-Schwarz inequality,

$$R_1(h) \leq \left(\int_0^{B^2} \Theta(h, t, \mathcal{X})\, dt\right)^{1/2} \left(\int_0^{B^2} 1\, dt\right)^{1/2} = B \left(\int_0^{B^2} \Theta(h, t, \mathcal{X})\, dt\right)^{1/2}. \quad (6)$$

Let $D$ denote the uniform probability distribution associated to the sample $\mathcal{X}$. Thus, $D(x) = \frac{1}{m+u}$ for all $x \in \mathcal{X}$. Let $\Pr_{x \sim D}[\mathcal{E}_x]$ denote the probability of event $\mathcal{E}_x$ when $x$ is randomly drawn according to $D$. By definition of $R_0$ and the Lebesgue integral, for all $h \in H$,

$$R_0(h) = \int_{\mathcal{X}} (h(x) - y_x)^2 D(x)\, dx = \int_0^\infty \Pr_{x \sim D}[(h(x) - y_x)^2 > t]\, dt = \int_0^{B^2} \Theta(h, t, \mathcal{X})\, dt. \quad (7)$$

Similarly, setting $\mathcal{X}_m = \{x_i \in \mathcal{X} : i \in [1, m]\}$ and $\mathcal{X}_u = \{x_i \in \mathcal{X} : i \in [m+1, m+u]\}$, we have

$$\widehat{R}(h) = \int_0^{B^2} \Theta(h, t, \mathcal{X}_m)\, dt \quad \text{and} \quad R(h) = \int_0^{B^2} \Theta(h, t, \mathcal{X}_u)\, dt. \quad (8)$$

In view of Equation 7, Inequality 6 can be rewritten as: $R_1(h) \leq B\sqrt{R_0(h)}$. By [Cortes and Mohri, 2006][Theorem 2], for all $\epsilon > 0$ and for any $t \geq 0$,

$$\Pr[\sup_{h \in H} \frac{\Theta(h, t, \mathcal{X}_u) - \Theta(h, t, \mathcal{X}_m)}{\sqrt{\Theta(h, t, \mathcal{X})}} > \epsilon] \leq \mathcal{N}(m + u)\bar{\Gamma}(\epsilon). \quad (9)$$

Fix $\epsilon > 0$. Then, with probability at least $1 - \mathcal{N}(m+u)\bar{\Gamma}(\epsilon)$, for all integers $n > 1$ and $i \geq 0$,

$$\frac{\Theta(h, \frac{iB^2}{n}, \mathcal{X}_u) - \Theta(h, \frac{iB^2}{n}, \mathcal{X}_m)}{\sqrt{\Theta(h, \frac{iB^2}{n}, \mathcal{X})}} \leq \epsilon. \tag{10}$$

Then, the convergence of the Riemann sums to the integral ensures that

$$R(h) - \widehat{R}(h) = \lim_{n \to \infty} \frac{1}{n} \sum_{i=0}^{n} \Theta(h, \frac{iB^2}{n}, \mathcal{X}_u) - \frac{1}{n} \sum_{i=0}^{n} \Theta(h, \frac{iB^2}{n}, \mathcal{X}_m) \tag{11}$$

$$\leq \epsilon \lim_{n \to \infty} \frac{1}{n} \sum_{i=0}^{n} \sqrt{\Theta(h, \frac{iB^2}{n}, \mathcal{X})} = \epsilon R_1(h) \leq \epsilon B \sqrt{R_0(h)}. \tag{12}$$

Let $\delta > 0$ and select $\epsilon = \epsilon_0$ as the minimum value of $\epsilon$ such that $\mathcal{N}(m+u)\bar{\Gamma}(\epsilon) \leq \delta$, then with probability at least $1 - \delta$,

$$R(h) - \widehat{R}(h) \leq \epsilon_0 B \sqrt{R_0(h)}. \tag{13}$$

Plugging in the following expression of $R_0(h)$ with respect to $R(h)$ and $\widehat{R}(h)$

$$R_0(h) = \frac{m}{m+u} \widehat{R}(h) + \frac{u}{m+u} R(h), \tag{14}$$

and solving the second-degree equation in $R(h)$ yields directly the statement of the theorem. □

Theorem 1 provides a general bound on the regression error within the transduction setting. The theorem can also be used to derive a bound in the classification case by simply setting $B = 1$. The resulting bound coincides with the tight classification bound given by Vapnik [1998]. The bound given by Theorem 1 depends on the function $\bar{\Gamma}$ and is implicit. The following provides a general and *explicit* error bound for transduction regression directly expressed in terms of the empirical error, the number of equivalence $\mathcal{N}(m+u)$ or the VC-dimension $d$, and the sample sizes $m$ and $u$.

**Corollary 1** *Let $H$ be a set of hypotheses with VC-dimension $d$. Assume that the loss function is bounded: for all $h \in H$ and $x \in \mathcal{X}$, $(h(x) - y_x)^2 \leq B^2$, where $B \in \mathbb{R}_+$. Then, with probability at least $1 - \delta$, for all $h \in H$,*

$$R(h) \leq \widehat{R}(h) + \frac{u\alpha^2 B^2}{2(m+u)} + \alpha B \sqrt{\widehat{R}(h) + \left(\frac{u\alpha B}{2(m+u)}\right)^2}, \tag{15}$$

*with $\alpha = \sqrt{\frac{2(m+u)}{mu}\left(\log \mathcal{N}(m+u) + \log\frac{1}{\delta}\right)} \leq \sqrt{\frac{2(m+u)}{mu}\left(d\log\frac{(m+u)e}{d} + \log\frac{1}{\delta}\right)}$.*

*Proof.* By Theorem 1, Inequality 15 holds for all $\alpha > 0$ such that $\mathcal{N}(m+u)\bar{\Gamma}(\alpha) \leq \delta$. By [Cortes and Mohri, 2006][Corollary 2], $\log\left(\mathcal{N}(m+u)\bar{\Gamma}(\alpha)\right) \leq \log \mathcal{N}(m+u) - \frac{1}{2}\frac{mu}{m+u}\alpha^2$. Setting $\log \delta$ to match this upper bound yields the expression of $\alpha$ given above. Since $\mathcal{N}(m+u)$ is bounded by the shattering coefficient of $H$ of order $m + u$, by Sauer's lemma, $\log \mathcal{N}(m+u) \leq d\log\frac{(m+u)e}{d}$. This gives the upper bound on $\alpha$ in terms of the VC-dimension. □

The bound is explicit and can be readily used within the Structural Risk Minimization (SRM) framework, either by using the expression of $\alpha$ in terms of the VC-dimension, or the tighter expression with respect to the number of equivalence classes $\mathcal{N}$. In the latter case, a structure of increasing number of equivalence classes can be constructed as in [Vapnik, 1998, page 360]. A more practical algorithm inspired by these concepts is described in the next section.

## 4 Transductive Regression Algorithm

This section presents an algorithm for the transductive regression problem.

Before presenting this algorithm, let us first emphasize that the algorithms introduced for transductive classification problems, e.g., transductive SVMs [Vapnik, 1998, Joachims, 1999], cannot be readily used for regression. These algorithms typically select the hypothesis $h$, out of a hypothesis space $H$, that minimizes the following optimization function

$$\min_{y^*_{m+i}, i=1,\ldots,u} \Omega(h) + C\frac{1}{m}\sum_{i=1}^{m} L\left(h(x_i), y_i\right) + C'\frac{1}{u}\sum_{i=1}^{u} L\left(h(x_{m+i}), y^*_{m+i}\right), \tag{16}$$

where $\Omega(h)$ is a capacity measure term, $L$ is the loss function used, $C \geq 0$ and $C' \geq 0$ regularization parameters, and where the minimum is taken over all possible labels $y^*_{m+1}, \ldots, y^*_{m+u}$ for the test points. In regression, this scheme would lead to a trivial solution not exploiting the transduction setting. Indeed, let $h_0$ be the hypothesis minimizing the first two terms, that is the solution of the induction problem. For the particular choice $y^*_{m+i} = h_0(x_{m+i})$, $i = 1, \ldots, u$, the third term vanishes. Thus, $h_0$ is also minimizing the sum of all three terms. In two-group classification, the trivial solution is typically not the solution of the minimization problem because in general $h_0(x_{m+i})$ is not in $\{0, 1\}$.

The main idea behind the design of our algorithm is to exploit the additional information provided in transduction, that is the position of the unlabeled examples. Our algorithm has two stages. The first stage is based on the position of unlabeled points. For each unlabeled point $x_i$, $i = m+1, \ldots, m+u$, a local estimate label $\bar{y}_i$ is determined using the labeled points in the neighborhood of $x_i$. In the second stage, a global hypothesis $h$ is found that best matches all labels, those of the training data and the estimate labels $\bar{y}_i$.

This second stage is critical and distinguishes our method from other suggested ones. While using local information to determine labels is important (see for example the discussion of Vapnik [1998]), it is not sufficient for a robust prediction. A global estimate of all labels is needed to make predictions less vulnerable to noise.

## 4.1   Local Estimates

Let $\Phi$ be a feature mapping from $\mathcal{X}$ to a vector space $F$ provided with a norm. We fix a radius $r \geq 0$ and consider for all $x' \in \mathcal{X}_u$, the ball of radius $r$ centered in $\Phi(x')$, denoted by $\mathcal{B}(\Phi(x'), r)$. This defines the neighborhood of the image of each unlabeled point. A single radius $r$ is used for all neighborhoods to limit the number of parameters for the algorithm. Labeled points $x \in \mathcal{X}_m$ whose images $\Phi(x)$ fall within the neighborhood of $\Phi(x')$, $x' \in \mathcal{X}_u$, help determine an estimate label of $x'$.

With a very large radius $r$, the labels of all training examples contribute to the definition of the local estimates. But, with smaller radii, only a limited number of computations are needed. When no such labeled point exists in the neighborhood of $x' \in \mathcal{X}_u$, which depends on the radius $r$ selected, $x'$ is disregarded in both training stages of the algorithm.

There are many possible ways to define the estimate label of $x' \in \mathcal{X}_u$ based on the neighborhood points. One simple way consists of defining it as the weighted average of the neighborhood labels $y_x$, where the weights may be defined as the inverse of distances of $\Phi(x)$ to $\Phi(x')$, or as similarity measures $K(x, x')$ when a positive definite kernel $K$ is associated to $\Phi$. Thus, when the set of labeled points with images in the neighborhood of $\Phi(x')$ is not empty, $I = \{i \in [1, m] : \Phi(x_i) \in \mathcal{B}(\Phi(x'), r)\} \neq \emptyset$, the estimate label $\bar{y}_{x'}$ of $x' \in \mathcal{X}_u$ can be given by:

$$\bar{y}_{x'} = \sum_{i \in I} \frac{w_i \bar{y}_i}{\sum_i w_i} \quad \text{with} \quad w_i^{-1} = \|\Phi(x') - \Phi(x_i)\| \leq r \quad \text{or} \quad w_i = K(x', x_i). \tag{17}$$

The estimate labels can also be obtained as the solution of a local linear or kernel ridge regression, which is what we used in most of our experiments.

In practice, with a relatively small radius $r$, the computation of an estimated label $\bar{y}_i$ depends only on a limited number of labeled points and their labels, and is quite efficient.

## 4.2   Global Optimization

The second stage of our algorithm consists of selecting a hypothesis $h$ that fits best the labels of the training points and the estimate labels provided in the first stage. As suggested by Corollary 1, hypothesis spaces with a smaller number of equivalence classes guarantee a better generalization error. The bound also suggests reducing the empirical error. This leads us to consider the following objective function

$$G = \|w\|^2 + C \sum_{i=1}^{m} (h(x_i) - y_i)^2 + C' \sum_{i=m+1}^{m+u} (h(x_i) - \bar{y}_i)^2, \tag{18}$$

where $h$ is as a linear function with weight vector $w \in F$: $\forall x \in \mathcal{X}, h(x) = w \cdot \Phi(x)$, and where $C \geq 0$ and $C' \geq 0$ are regularization parameters. The first two terms of the objective function coincide with those used in standard (kernel) ridge regression. The third term, which restricts the estimate error, can be viewed as imposing a smaller number of equivalence classes on the hypothesis space as suggested by the error bound of Corollary 1. The constraint explicitly exploits knowledge

about the location of all the test points, and limits the range of the hypothesis at these locations, thereby reducing the number of equivalence classes. Our algorithm can be viewed as a generalization of (kernel) ridge regression to the transductive setting. In the following, we will show that this generalized optimization problem admits a closed-form solution and a natural kernel-based solution.

### 4.2.1 Primal solution

Let $N$ be the dimension of the feature space and let $\mathbf{W} \in \mathbb{R}^{N \times 1}$ denote the column matrix whose components are the coordinates of $w$, $\mathbf{Y} \in \mathbb{R}^{m \times 1}$ the column matrix whose components are the labels $y_i$ of the training examples, and $\mathbf{Y}' \in \mathbb{R}^{u \times 1}$ the column-matrix whose components are the estimated labels $\bar{y}_i$ of the test examples. Let $\mathbf{X} = [\Phi(x_1), \ldots, \Phi(x_m)] \in \mathbb{R}^{N \times m}$ denote the matrix whose columns are the components of the images by $\Phi$ of the training examples, and similarly $\mathbf{X}' = [\Phi(x_{m+1}), \ldots, \Phi(x_{m+u})] \in \mathbb{R}^{N \times u}$ the matrix corresponding to the test examples. $G$ can then be rewritten as:

$$G = \|\mathbf{W}\|^2 + C\|\mathbf{X}^\top \mathbf{W} - \mathbf{Y}\|^2 + C'\|\mathbf{X}'^\top \mathbf{W} - \mathbf{Y}'\|^2. \tag{19}$$

$G$ is convex and differentiable and its gradient is given by

$$\nabla G = 2\mathbf{W} + 2C\,\mathbf{X}(\mathbf{X}^\top \mathbf{W} - \mathbf{Y}) + 2C'\,\mathbf{X}'(\mathbf{X}'^\top \mathbf{W} - \mathbf{Y}'). \tag{20}$$

The matrix $\mathbf{W}$ minimizing $G$ is the unique solution of $\nabla G = 0$. Since $(\mathbf{I}_N + C\,\mathbf{X}\mathbf{X}^\top + C'\,\mathbf{X}'\mathbf{X}'^\top)$ is invertible, it is given by the following expression

$$\mathbf{W} = (\mathbf{I}_N + C\,\mathbf{X}\mathbf{X}^\top + C'\,\mathbf{X}'\mathbf{X}'^\top)^{-1}(C\,\mathbf{X}\mathbf{Y} + C'\,\mathbf{X}'\mathbf{Y}'). \tag{21}$$

This gives a closed-form solution in the primal space based on the inversion of a matrix in $\mathbb{R}^{N \times N}$. Let $T(N)$ be the time complexity of computing the inverse of a matrix in $\mathbb{R}^{N \times N}$. $T(N) = O(N^3)$ using standard methods or $T(N) = O(N^{2.376})$ with the method of Coppersmith and Winograd. The time complexity of the computation of $\mathbf{W}$ from $\mathbf{X}, \mathbf{X}', \mathbf{Y}$, and $\mathbf{Y}'$ is thus in $O(T(N) + (m+u)N^2)$.

When the dimension $N$ of the feature space is small compared to the number of examples $m + u$, which is typical in modern learning applications where $u$ is large, this method remains practical and leads to a very efficient computation. The use of the so-called *empirical kernel map* [Schölkopf and Smola, 2002] also makes this method very attractive. Given a kernel $K$, the empirical kernel feature vector associated to $x$ is the $m$-dimensional vector $\Phi(x) = [K(x, x_1), \ldots, K(x, x_m)]^\top$. Thus, the dimension of the feature space is then $N = m$. For relatively small $m$, even for very large values of $u$ with respect to $m$, the solution is efficiently computable and yet benefits from the use of kernels.

This computational advantage is not shared by other methods such as the manifold regularization techniques [Belkin et al., 2004], or even by the regression technique described by [Chapelle et al., 1999], despite it is based on a primal method (we have derived a dual version of that method as well, see Section 5) since it requires among other things the inversion of a matrix in $\mathbb{R}^{u \times u}$.

Once $\mathbf{W}$ is computed, prediction can be done by computing $\mathbf{X}'^\top \mathbf{W}$ in time $O(uN)$.

### 4.2.2 Dual solution

The computation can also be done in the dual space, which is useful in the case of very high-dimensional feature spaces. Let $\mathbf{M}_X \in \mathbb{R}^{N \times (m+u)}$ and $\mathbf{M}_Y \in \mathbb{R}^{(m+u) \times 1}$ be the matrices defined by:

$$\mathbf{M}_X = \begin{pmatrix} \sqrt{C}\,\mathbf{X} & \sqrt{C'}\,\mathbf{X}' \end{pmatrix} \quad \mathbf{M}_Y = \begin{pmatrix} \sqrt{C}\,\mathbf{Y} \\ \sqrt{C'}\,\mathbf{Y}' \end{pmatrix}. \tag{22}$$

Then, Equation 21 can be rewritten as: $\mathbf{W} = (\mathbf{I}_N + \mathbf{M}_X\mathbf{M}_X^\top)^{-1}\mathbf{M}_X\mathbf{M}_Y$. To determine the dual solution, observe that

$$\mathbf{M}_X^\top(\mathbf{M}_X\mathbf{M}_X^\top + \gamma\mathbf{I}_N)^{-1} = (\mathbf{M}_X^\top\mathbf{M}_X + \gamma\mathbf{I}_{m+u})^{-1}\mathbf{M}_X^\top, \tag{23}$$

where $\mathbf{I}_{m+u}$ denotes the identity matrix of $\mathbb{R}^{(m+u) \times (m+u)}$. This can be derived without difficulty from a series expansion of $(\mathbf{M}_X\mathbf{M}_X^\top + \gamma\mathbf{I}_N)^{-1}$. Thus, $\mathbf{W}$ can also be computed via:

$$\mathbf{W} = \mathbf{M}_X(\mathbf{I}_{m+u} + \mathbf{K})^{-1}\mathbf{M}_Y, \tag{24}$$

where $\mathbf{K}$ is the Gram matrix $\mathbf{K} = \mathbf{M}_X^\top\mathbf{M}_X$. Let $\mathbf{K}_{21} \in \mathbb{R}^{u \times m}$ and $\mathbf{K}_{22} \in \mathbb{R}^{u \times u}$ be the sub-matrices of the Gram $\mathbf{K}$ defined by: $\mathbf{K}_{21} = (K(x_{m+i}, x_j)_{1 \leq i \leq u, 1 \leq j \leq m})$ and $\mathbf{K}_{22} = (K(x_{m+i}, x_{m+j})_{1 \leq i,j \leq u})$ and let $\mathbf{K}_2 \in \mathbb{R}^{u \times (m+u)}$ be the matrix defined by:

$$\mathbf{K}_2 = \begin{pmatrix} \sqrt{C}\,\mathbf{K}_{21} & \sqrt{C'}\,\mathbf{K}_{22} \end{pmatrix} = \mathbf{X}'^\top\mathbf{M}_X. \tag{25}$$

| Dataset | No. of unlab. points | Relative improvement in MSE (%) | | |
|---|---|---|---|---|
| | | Our algorithm | Chapelle et al. [1999] | Belkin et al. [2004] |
| Boston Housing [13] | 25 | 20.2±14.7 | 4.3±11.3 | 2.4±5.4 |
| California Housing [8] | 500 | 8.4±6.9 | 2.7±3.0 | 3.9±12.3 |
| | 2,500 | 25.9±8.3 | 0.2±0.3 | 0.0±0.0 |
| | 5,000 | 17.2±8.7 | 0.0±0.0 | 0.0±0.0 |
| | 20,000 | 22.0±11.0 | — | — |
| kin-32fh [32] | 2,500 | 9.4±3.7 | 2.2±2.6 | 2.7±3.1 |
| | 8,000 | 18.4±5.9 | 0.5±0.5 | 0.9±0.7 |
| Elevators [18] | 500 | 14.4±10.4 | 1.5±2.7 | 2.6 ±7.7 |
| | 2500 | 9.0±6.9 | 2.2±2.9 | 0.0±0.0 |
| | 15,000 | 9.7±5.8 | — | — |

Table 1: Transductive regression experiments. The number in brackets after the name indicates the input dimensionality of the data set. The number of training examples was $m = 481$ for the Boston Housing data set, $m = 25$ for the other tasks. The number of unlabeled examples was $u = 25$ for the Boston Housing data set and varied from $u = 500$ to the maximum of 20,000 examples for the California Housing data set. For $u \geq 10,000$, the algorithms of Chapelle et al. [1999] and Belkin et al. [2004] did not terminate within the time period of our experiments.

Then, predictions can be made using kernel functions alone since $X'^\top W$ can be computed by:

$$\mathbf{X}'^\top \mathbf{W} = \mathbf{X}'^\top \mathbf{M}_X (\mathbf{I}_{m+u} + \mathbf{K})^{-1} \mathbf{M}_Y = \mathbf{K}_2 (\mathbf{I}_{m+u} + \mathbf{K})^{-1} \mathbf{M}_Y. \tag{26}$$

When the dimension of the feature space $N$ is very large with respect to the total number of examples, this can lead to a faster computation of the solution. $(\mathbf{I}_{m+u} + \mathbf{K})^{-1} \mathbf{M}_Y$ can be computed in $O(T(m + u) + (m + u)^2 t_K)$ and predictions are computed in time $O(u\,(m + u))$, where $t_K$ is the time complexity of the computation of $K(x, x)$, $x, x' \in \mathcal{X}$. As already pointed in the description of the local estimates, in practice, some unlabeled points are disregarded in the training phases because no labeled point falls in their neighborhood. Thus, instead of $u$, a smaller number of unlabeled examples $u' \leq u$ determines the computational cost.

## 5 Experimental Results

This section reports the results of our experiments with the transductive regression algorithm just presented with several data sets. For comparison, we also implemented the algorithm of Chapelle et al. [1999] and that of Belkin et al. [2004], which are among the very few algorithms described in the literature dealing specifically with the problem of transductive regression. For the algorithm of Chapelle et al. [1999], we in fact derived and implemented a dual solution not described in the original paper. With the notation used in that paper, it can be shown that

$$\mathbf{C} = \mathbf{I} - \hat{\mathbf{K}}\hat{\mathbf{K}}^\top (\hat{\mathbf{K}}\hat{\mathbf{K}}^\top + \gamma \mathbf{I})^{-1}. \tag{27}$$

Our comparisons were made using several publicly available regression data sets: *Boston Housing*, *kin-32fh* a data set in the *Kinematics* family with high unpredictability or noise, *California Housing*, and *Elevators* [Torgo, 2006]. For the Boston Housing data set, we used the same partitioning of the training and test sets as in [Chapelle et al., 1999]: 481 training examples and 25 test examples. The input variables were normalized to have mean zero and a variance one. For the kin-32fh, California Housing, and Elevators data sets, 25 training examples were used with varying (large) amounts of test examples: 2,500 and 8,000 for kin-32fh; from 500 up to 20,000 for California Housing; and from 500 to 15,000 for Elevators. The experiments were repeated for 100 random partitions of training and test sets.

The kernels used with all algorithms were Gaussian kernels. To measure the improvement produced by the transductive inference algorithms, we used kernel ridge regression as a baseline. The optimal values for the width of the Gaussian $\sigma$ and the ridge $\frac{1}{C}$ were determined using cross-validation. These parameters were then fixed at these values. The remaining parameters for our algorithm, $r$ and $C'$, were determined using a grid search and cross-validation. The parameters of the algorithms of Chapelle et al. [1999] and Belkin et al. [2004] were determined in the same way. Alternatively, the parameters could be selected using the explicit VC-dimension generalization bound of Corollary 1. For our algorithm, we found the best values of $r$ to be typically among the 2.5% smallest distances between training and test points. Thus, each estimate label was determined by only a small number of labeled points.

For our algorithm, we experimented both with the dual solution using Gaussian kernels, and the primal solution with an empirical Gaussian kernel map as described in Section 4.2.1. The results

obtained were very similar, however the primal method was dramatically faster since it required the inversion of relatively small-dimensional matrices even for a large number of unlabeled examples. For consistency, all the results reported for our method relate to the dual solution, except from those with very large $u$, e.g. $u \leq 10{,}000$, where the dual method was too time-consuming.

Table 1 shows the results of our experiments. For each data set and each algorithm, the relative improvement in mean squared error (MSE) with respect to the baseline averaged over the random partitions is indicated, followed by its standard deviation. Some improvements were small or not statistically significant. In general, we observed no significant performance improvement over the baseline on any of these data sets using the Laplacian regularized least squares method of Belkin et al. [2004]. We note that, while positive classification results have been previously reported for this algorithm, no transductive regression experimental result seems to have been published for it. Our results for the method of Chapelle et al. [1999] match those reported by the authors for the Boston Housing data set (both absolute and relative MSE).

Our algorithm achieved a significant improvement of the MSE in all data sets and for different amounts of unlabeled data and was shown to be practical for large data sets of 20,000 test examples. This matches many real-world situations where amount of unlabeled data is orders of magnitude larger than that of labeled data.

## 6 Conclusion

We presented a general study of transductive regression. We gave new and general explicit error bounds for transductive regression and described a simple and general algorithm inspired by our bound that can scale to relatively large data sets. The results of experiments show that our algorithm achieves a smaller error in several tasks compared to other previously published algorithms for transductive regression.

The problem of transductive regression arises in a variety of learning contexts, in particular for learning node labels of a very large graphs such as the web graph. This leads to computational problems that may require approximations or new algorithms. We hope that our study will be useful for dealing with these and other similar transduction regression problems.

## Footnotes

[1]This is in fact one of the two transduction settings discussed by [Vapnik, 1998], but, under some general conditions, the results proved with this setting carry over to the other.

## References

Mikhail Belkin, Partha Niyogi, and Vikas Sindhwani. Manifold regularization; a geometric framework for learning from examples. Technical Report TR-2004-06, University of Chicago, 2004.

Kristin Bennett and Ayhan Demiriz. Semi-supervised support vector machines. *NIPS 11*, pages 368–374, 1998.

Olivier Chapelle, Vladimir Vapnik, and Jason Weston. Transductive Inference for Estimating Values of Functions. *NIPS 12*, pages 421–427, 1999.

Adrian Corduneanu and Tommi Jaakkola. On information regularization. In Christopher Meek and Uffe Kjærulff, editors, *Proceedings of the Nineteenth Annual Conference on Uncertainty in Artificial Intelligence*, pages 151–158, 2003.

Corinna Cortes and Mehryar Mohri. On Transductive Regression. Technical Report TR2006-883, Courant Institute of Mathematical Sciences, New York University, November 2006.

Philip Derbeko, Ran El-Yaniv, and Ron Meir. Explicit learning curves for transduction and application to clustering and compression algorithms. *J. Artif. Intell. Res. (JAIR)*, 22:117–142, 2004.

Thore Graepel, Ralf Herbrich, and Klaus Obermayer. Bayesian transduction. *NIPS 12*, 1999.

Thorsten Joachims. Transductive inference for text classification using support vector machines. In Ivan Bratko and Saso Dzeroski, editors, *Proceedings of ICML-99, 16th International Conference on Machine Learning*, pages 200–209. Morgan Kaufmann Publishers, San Francisco, US, 1999.

Gert R. G. Lanckriet, Nello Cristianini, Peter Bartlett, Laurent El Ghaoui, and Michael I. Jordan. Learning the kernel matrix with semidefinite programming. *J. Mach. Learn. Res.*, 5:27–72, 2004. ISSN 1533-7928.

Bernhard Schölkopf and Alex Smola. *Learning with Kernels*. MIT Press: Cambridge, MA, 2002.

Dale Schuurmans and Finnegan Southey. Metric-Based Methods for Adaptive Model Selection and Regularization. *Machine Learning*, 48:51–84, 2002.

Luís Torgo. Regression datasets, 2006. http://www.liacc.up.pt/ ltorgo/Regression/DataSets.html.

Vladimir N. Vapnik. *Estimation of Dependences Based on Empirical Data*. Springer, Berlin, 1982.

Vladimir N. Vapnik. *Statistical Learning Theory*. Wiley-Interscience, New York, 1998.

Dengyong Zhou, Jiayuan Huang, and Bernard Scholkopf. Learning from labeled and unlabeled data on a directed graph. In L. De Raedt and S. Wrobel, editors, *Proceedings of ICML-05*, pages 1041–1048, 2005.

Xiaojin Zhu, Jaz Kandola, Zoubin Ghahramani, and John Lafferty. Nonparametric transforms of graph kernels for semi-supervised learning. *NIPS 17*, 2004.
